# Spectro-Temporal Receptive Fields of Subthreshold Responses in Auditory Cortex

**Christian K. Machens, Michael Wehr, Anthony M. Zador**
Cold Spring Harbor Laboratory
One Bungtown Rd
Cold Spring Harbor, NY 11724
{machens, wehr, zador}@cshl.edu

## Abstract

How do cortical neurons represent the acoustic environment? This question is often addressed by probing with simple stimuli such as clicks or tone pips. Such stimuli have the advantage of yielding easily interpreted answers, but have the disadvantage that they may fail to uncover complex or higher-order neuronal response properties.

Here we adopt an alternative approach, probing neuronal responses with complex acoustic stimuli, including animal vocalizations and music. We have used *in vivo* whole cell methods in the rat auditory cortex to record subthreshold membrane potential fluctuations elicited by these stimuli. Whole cell recording reveals the total synaptic input to a neuron from all the other neurons in the circuit, instead of just its output—a sparse binary spike train—as in conventional single unit physiological recordings. Whole cell recording thus provides a much richer source of information about the neuron's response.

Many neurons responded robustly and reliably to the complex stimuli in our ensemble. Here we analyze the linear component—the *spectro-temporal receptive field* (STRF)—of the transformation from the sound (as represented by its time-varying spectrogram) to the neuron's membrane potential. We find that the STRF has a rich dynamical structure, including excitatory regions positioned in general accord with the prediction of the simple tuning curve. We also find that in many cases, much of the neuron's response, although deterministically related to the stimulus, cannot be predicted by the linear component, indicating the presence of as-yet-uncharacterized nonlinear response properties.

## 1 Introduction

In their natural environment, animals encounter highly complex, dynamically changing stimuli. The auditory cortex evolved to process such complex sounds. To investigate a system in its normal mode of operation, it therefore seems reasonable to use natural stimuli.

The linear response of an auditory neuron can be described in terms of its spectro-temporal receptive field (STRF). The cortical STRF has been estimated using a variety of stimu-

lus ensembles[1], including tone pips [1] and dynamic ripples [2]. However, while natural stimuli have long been used to probe cortical responses [3, 4], and have been widely used in other preparations to compute STRFs [5], they have only rarely been used to compute STRFs from cortical neurons [6].

Here we present estimates of the STRF using *in vivo* whole cell recording. Because whole cell recording measures the total synaptic input to a neuron, rather than just its output— a sparse binary spike train—as in conventional single unit physiological recordings, this technique provides a much richer source of information about the neuron's response.

Whole cell recording also has a different sampling bias from conventional extracellular recording: instead of recording from active neurons with large action potentials (*i.e.* those that are most easily isolated on the electrode), whole cell recording selects for neurons solely on the basis of the experimenter's ability to form a gigaohm seal.

Using these novel methods, we investigated the computations performed by single neurons in the auditory cortex A1 of rats.

## 2   Spike responses and subthreshold activity

We first used cell-attached methods to obtain well-isolated single unit recordings. We found that many cells in auditory cortex responded only very rarely to the natural stimulus ensemble, making it difficult to characterize the neuron's input-output relationship effectively. An example of this problem is shown in Fig. 1(b) where a natural stimulus (here, the call of a nightingale) leads to an average of about five spikes during the eight-second-long presentation. Such sparse responses are not surprising, since it is well known that many cortical neurons are selective for stimulus transients [7, 8].

One way to circumvent this difficulty is to present stimuli that elicit high firing rates. For example, using dynamic ripple stimuli, an STRF can be constructed with about $10,000$ spikes collected over 20 minutes (average firing rate of approximately 8 spikes/second, or about 10-fold higher than the rate elicited by the natural stimulus in Fig. 1(b)) [9]. However, such stimuli have, by design, a simple correlational structure, and therefore preclude the investigation of nonlinear response properties driven by higher-order stimulus characteristics.

We have therefore adopted an alternative approach based on *in vivo* whole cell recording, exploiting the fact that although these neurons spike only rarely, they feature strong subthreshold activity. A set of subthreshold voltage traces, obtained by a whole-cell recording where spikes were blocked (only in the neuron being recorded from) with the intracellular sodium channel blocker QX-314 (*see Methods*), is shown in Fig. 1(c). The responses feature robust stimulus-locked fluctuations of membrane potential, as well as some spontaneous activity. Both the spontaneous and stimulus-locked voltage fluctuations are due to the synchronous arrival of many excitatory postsynaptic potentials (EPSPs). (Note that if spikes had not been blocked pharmacologically, some of the larger EPSPs would have triggered spikes). Not only do these whole cell recordings avoid the problem of sparse spiking responses, they also provide insight into the computations performed by the input to the neuron's spike generating mechanism.

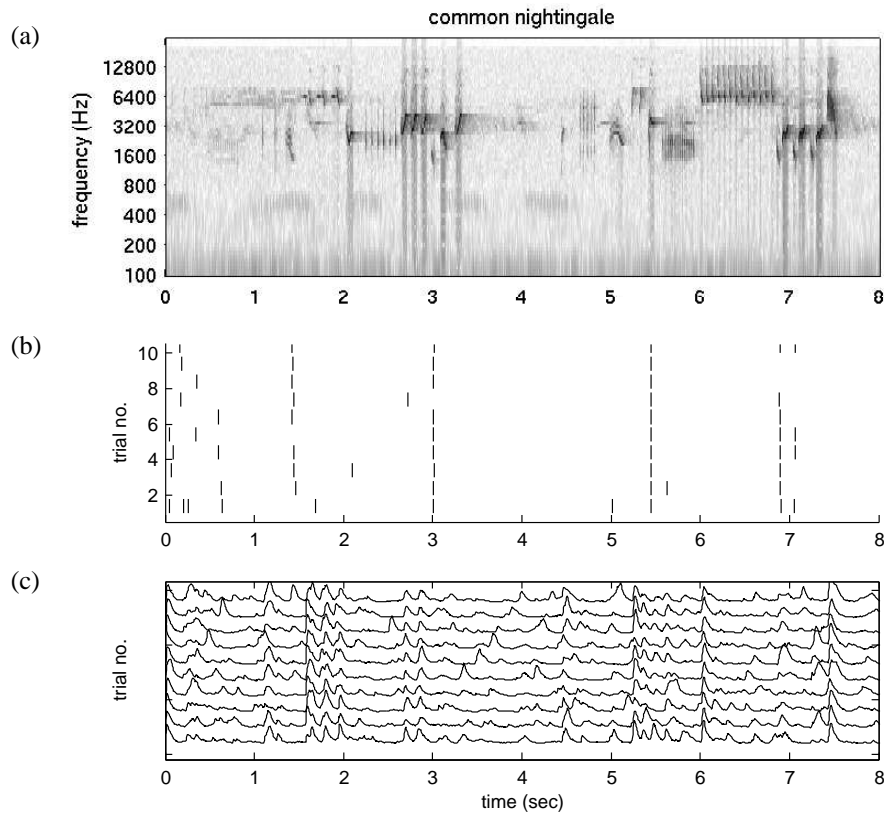

Figure 1: (a) Spectrogram of the song of a nightingale. (b) Spike raster plots recorded in cell-attached mode during ten repetitions of the nightingale song from a single neuron in auditory cortex A1. (c) Voltage traces recorded in whole-cell-mode during ten repetitions from another neuron in A1.

## 3   Reliability of responses

A key step in the characterization of the neuron's responses is the separation of the stimulus-locked activity from the stimulus-independent activity ("background noise"). A sample average trace is compared with a single trial in Fig. 2(a).

To quantify the amount of stimulus-locked activity, we computed the coherence function between a single response trace and the average over the remaining traces. The coherence measures the frequency-resolved correlation of two time series. This function is shown in Fig. 2b for responses to several natural stimuli from the same cell. The coherence function demonstrates that the stimulus-dependent activity is confined to lower frequencies ($< 40$ Hz). Note that the coherence function provides merely an average over the complete trace; in reality, the coherence can locally be much higher (when all traces feature the same stimulus-locked excursion in membrane potential) or much lower (for instance in the absence of stimulus-locked activity). On average, however, the coherence is approximately the same for all the natural stimuli presented, indicating that all stimuli feature approximately the same level of background activity.

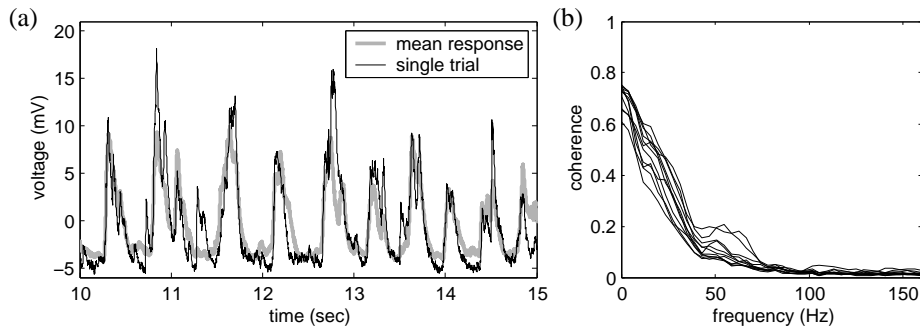

Figure 2: (a) Mean response compared to single trial for a natural stimulus (jaguar mating call). (b) Coherence functions between mean response and single trial for different stimuli. All natural stimuli yield approximately the same relation between signal and noise.

## 4   Spectro-temporal receptive field

Having established the mean over trials as a reliable estimate of the stimulus-dependent activity, we next sought to understand the computations performed by the neurons.

To mimic the cochlear transform, it has proven useful to describe the stimulus in the time-frequency domain [2]. Discretizing both time and frequency, we describe the stimulus power in the $i$-th time bin $t_i$ and the $k$-th frequency bin $f_k$ by $s(t_i, f_k)$. To compute the time-frequency representation, we used the spectrogram method which requires a certain choice for the time-frequency tradeoff [10]; several choices were used independently of each other, essentially yielding the same results. In all cases, stimulus power is measured in logarithmic units.

The simplest and most widely used model is a linear transform between the stimulus (as represented by the spectrogram) and the response, given by the formula

$$r_{\text{est}}(t_i) = r_0 + \sum_{j,k} H(t_j, f_k) s(t_i - t_j, f_k) \tag{1}$$

where $r_0$ is a constant offset and the parameters $H(t_j, f_k)$ represent the spectro-temporal receptive field (STRF) of the neuron. Note, though, that the response is usually taken to be the average firing rate [2, 11]; here the response is given by the subthreshold voltage trace. The parameters can be fitted by minimizing the mean-square error between the measured response $r(t)$ and the estimated response $r_{\text{est}}(t)$. This problem is solved by multi-dimensional linear regression.

However, a direct, "naive" estimate as obtained by the solution to the regression equations, will usually fail since the stimulus does not properly sample all dimensions in stimulus space. In general, this leads to strong overfitting of the poorly sampled dimensions and poor predictive power of the model. The overfitting can be seen in the noisy structure of the STRF shown in Fig. 3(a).

A simple alternative is to penalize the improperly sampled directions which can be done using ridge regression [12]. Ridge regression minimizes the mean-square-error between measured and estimated response while placing a constraint on the sum of the regression coefficients. Choosing the constraint such that the predictive power of the model is maximized, we obtained the STRF shown in Fig. 3(b). Note that ridge regression operates on all coefficients uniformly (*ie* the constraint is global), so that observed smoothness in the estimated STRF represents structure in the data; no local smoothness constraint was applied.

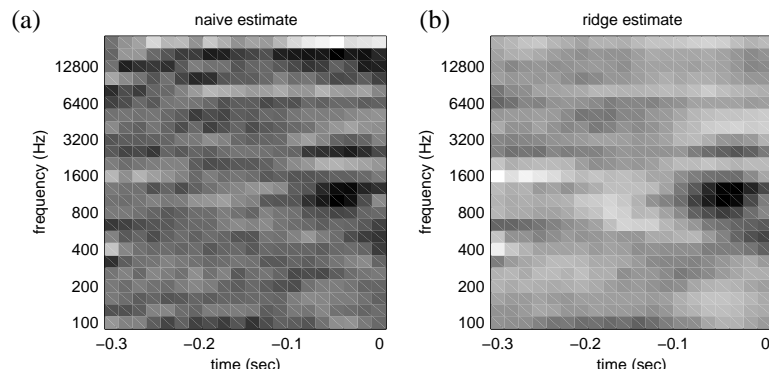

Figure 3: (a) Naive estimate of the STRF via linear regression. Darker pixels denote time-frequency bins with higher power. (b) Estimate of the STRF via ridge regression.

The STRF displays the neuron's frequency-sensitivity, centered around 800–1600 Hz. This range of frequencies matches the neuron's tuning curve which is measured with short sine tones. The STRF suggests that the neuron essentially integrates frequencies within this range and a time constant of about 100 ms. These types of computations have been previously reported for neurons in auditory cortex [1, 2].

## 4.1 Spectral analysis of error

How well does the simple linear model predict the subthreshold responses? To assess the predictive power of the model, the STRF was estimated from data obtained for ten different natural stimuli and then tested on an eleventh stimulus. A sample prediction is shown in Fig. 4(a). While the predicted trace roughly captures the occurrence of the EPSPs, it fails to predict their overall shape. This observation can be quantified by spectrally resolving the prediction success. For that purpose, we again used the coherence function which measures the correlation between the actual response and the predicted response at each frequency. This function is shown in Fig. 4(b). Clearly, the model fails to predict any response fluctuations faster than $\approx 5$ Hz. As a comparison, recall that the response is reliable up to about 40 Hz (Fig. 2).

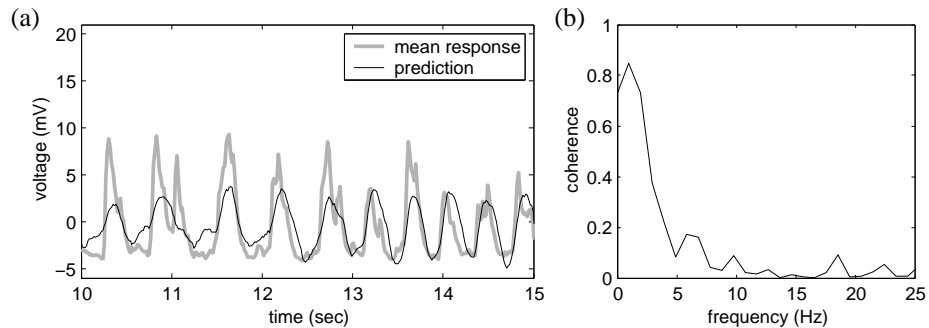

Figure 4: (a) Mean response and prediction for a natural stimulus (jaguar mating call). The STRF captures the gross features of the response, but not the fine details. (b) Coherence function between measured and predicted response.

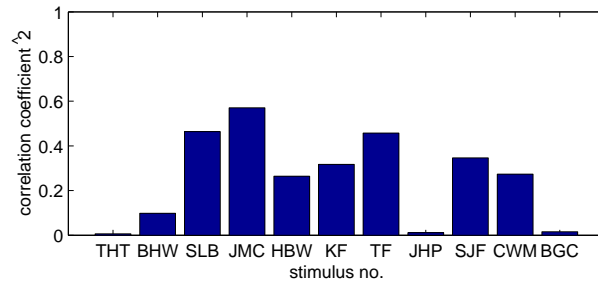

Figure 5: Squared Correlation coefficients between the mean of the measured responses and the predicted response. Linear prediction with the STRF is more effective for some stimuli than others.

### 4.2 Errors across stimuli

Some of the natural stimuli elicited highly reliable responses that were not at all predicted by the STRF, see Fig. 5. In fact, the example shown in Fig. 4 is one of the best predictions achieved by the model. The failure to predict the responses of some stimuli cannot be attributed to the absence of stimulus-locked activity; as the coherence functions in Fig. 2(a) have shown, all stimuli feature approximately the same proportion of stimulus-locked activity to noise. Rather, such responses indicate a high degree of nonlinearity that dominates the response to some stimuli. This observation is in accord with previous work on neurons in the auditory forebrain of zebrafinches [11], where neurons show a high degree of feature selectivity.

The nonlinearities seen in subthreshold responses of A1 neurons can partly be attributed to adaptation, to interactions between frequencies [13, 14], and also to *off-responses*[2]. In general, the linear model performs best if the stimuli are slowly modulated in both time and frequency.

## 5 Discussion

We have used whole cell patch clamp methods *in vivo* to record subthreshold membrane potential fluctuations elicited by natural sounds. Subthreshold responses were reliable and (in contrast to the suprathreshold spiking responses) sufficiently rich and robust to permit rapid and efficient estimation of the linear predictor of the neuron's response (the STRF). The present manuscript represents the first analysis of subthreshold responses elicited by natural stimuli in the cortex, or to our knowledge in any system.

STRFs estimated from natural sounds were in general agreement, with respect to gross characteristics such as frequency tuning, with those obtained directly from pure tone pips. The STRFs from complex sounds, however, provided a much more complete view of the neuron's dynamics, so that it was possible to compare the predicted and experimentally measured responses.

In many cases the prediction was poor (cf. Fig. 6), indicating strong nonlinearities in the neuron's responses. These nonlinearities include adaptation, two-tone interactions, and

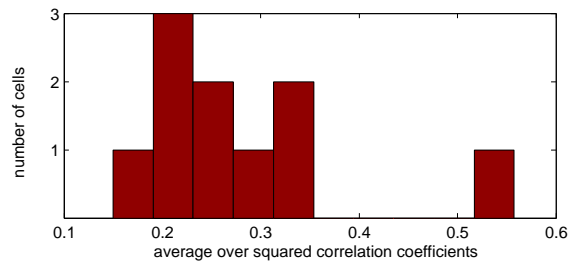

Figure 6: Summary figure. Altogether $n = 10$ cells were recorded in whole cell mode. Shown are the squared correlation coefficients, averaged over all stimuli for a given cell. For many cells, the linear model worked rather poorly as indicated by low cross correlations.

off-responses. Explaining these nonlinearities represents an exciting challenge for future research.

## 6   Methods

Sprague-Dawley rats (p18-21) were anesthetized with ketamine (30 mg/kg) and medetomidine (0.24 mg/kg). Whole cell recordings and single unit recordings were made with glass microelectrodes ($3 - 4$ M$\Omega$) from primary auditory cortex (A1) using standard methods appropriately modified for the *in vivo* preparation. During whole cell recordings, sodium action potentials were blocked using the sodium channel blocker QX-314.

All natural sounds were taken from an audio CD, sampled at 44,100 Hz. Animal vocalizations were from "The Diversity of Animal Sounds," available from the Cornell Laboratory of Ornithology. Additional stimuli included pure tones and white noise bursts with 25 ms duration and 5 ms ramp (sampled at 97.656 kHz), and *Purple Haze* by Jimi Hendrix. Sounds were delivered by a TDT RP2 at 97.656 kHz to a calibrated TDT electrostatic speaker and presented free field in a double-walled sound booth.

## Footnotes

[1]Because cortical neurons respond poorly to white noise, this stimulus has not been used to estimate cortical STRFs.

[2]Off-responses are excitatory responses that occur at the termination of stimuli in some neurons. Because they have the same sign as the on-response, they represent a form of rectifying nonlinearity. Further complications arise because on- and off-responses interact, depending on their spectro-temporal relations [14].

## References

[1] R. C. deCharms and M. M. Merzenich. Primary cortical representation of sounds by the coordination of action- potential timing. *Nature*, 381(6583):610–3., 1996.

[2] D. J. Klein, D. A. Depireux, J. Z. Simon, and S. A. Shamma. Robust spectrotemporal reverse correlation for the auditory system: optimizing stimulus design. *J Comput Neurosci*, 9(1):85–111., 2000.

[3] O. Creutzfeldt, F. C. Hellweg, and C. Schreiner. Thalamocortical transformation of responses to complex auditory stimuli. *Exp Brain Res*, 39(1):87–104, 1980.

[4] I. Nelken, Y. Rotman, and O. Bar Yosef. Responses of auditory-cortex neurons to structural features of natural sounds. *Nature*, 397:154–157, 1999.

[5] F. E. Theunissen, S. V. David, N. C. Singh, A. Hsu, W. E. Vinje, and J. L. Gallant. Estimating spatio-temporal receptive fields of auditory and visual neurons from their responses to natural stimuli. *Network*, 12(3):289–316., 2001.

[6] J. F. Linden, R. C. Liu, M. Kvale, C. E. Schreiner, and M. M. Merzenich. Reverse-correlation analysis of receptive fields in mouse and rat auditory cortex. *Society for Neuroscience Abstracts*, 27(2):1635, 2001.

[7]  P. Heil. Auditory cortical onset responses revisited. ii. response strength. *J Neurophysiol*, 77(5):2642–60., 1997.

[8]  S. L. Sally and J. B. Kelly. Organization of auditory cortex in the albino rat: sound frequency. *J Neurophysiol*, 59(5):1627–38., 1988.

[9]  D. A. Depireux, J. Z. Simon, D. J. Klein, and S. A. Shamma. Spectro-temporal response field characterization with dynamic ripples in ferret primary auditory cortex. *J Neurophysiol*, 85(3):1220–34., 2001.

[10]  L. Cohen. *Time-frequency Analysis*. Prentice Hall, 1995.

[11]  F. E. Theunissen, K. Sen, and A. J. Doupe. Spectral-temporal receptive fields of nonlinear auditory neurons obtained by using natural sounds. *J. Neurosci.*, 20(6):2315–2331, 2000.

[12]  T. Hastie, R. Tibshirani, and J. Friedman. *The elements of statistical learning theory.* Springer, 2001.

[13]  M. Brosch and C. E. Schreiner. Time course of forward masking tuning curves in cat primary auditory cortex. *J Neurophysiol*, 77(2):923–43., 1997.

[14]  L. Tai and A. Zador. In vivo whole cell recording of synaptic responses underlying two-tone interactions in rat auditory cortex. *Society for Neuroscience Abstracts*, 27(2):1634, 2001.
